# Kernel Hyperalignment

**Alexander Lorbert & Peter J. Ramadge**
Department of Electrical Engineering
Princeton University

## Abstract

We offer a regularized, kernel extension of the multi-set, orthogonal Procrustes problem, or hyperalignment. Our new method, called *Kernel Hyperalignment*, expands the scope of hyperalignment to include nonlinear measures of similarity and enables the alignment of multiple datasets with a large number of base features. With direct application to fMRI data analysis, kernel hyperalignment is well-suited for multi-subject alignment of large ROIs, including the entire cortex. We report experiments using real-world, multi-subject fMRI data.

## 1   Introduction

One of the goals of multi-set data analysis is forming qualitative comparisons between datasets. To the extent that we can control and design experiments to facilitate these comparisons, we must first ask whether the data are *aligned*. In its simplest form, the primary question of interest is whether corresponding features among the datasets measure the same quantity. If yes, we say the data are aligned; if not, we must first perform an alignment of the data.

The alignment problem is crucial to multi-subject fMRI data analysis, which is the motivation for this work. An appreciable amount of effort is devoted to designing experiments that maintain the focus of a subject. This is to ensure temporal alignment across subjects for a common stimulus. However, with each subject exhibiting his/her own unique spatial response patterns, there is a need for spatial alignment. Specifically, we want between subject correspondence of voxel $j$ at TR $i$ (Time of Repetition). The typical approach taken is anatomical alignment [20] whereby anatomical landmarks are used to anchor spatial commonality across subjects. In linear algebra parlance, anatomical alignment is an affine transformation with 9 degrees of freedom.

Recently, Haxby et al. [9] proposed *Hyperalignment*, a function-based alignment procedure. Instead of a 9-parameter transformation, a higher-order, orthogonal transformation is derived from voxel time-series data. The underlying assumption of hyperalignment is that, for a fixed stimulus, a subject's time-series data will possess a common geometry. Accordingly, the role of alignment is to find isometric transformations of the per-subject trajectories traced out in voxel space so that the transformed time-series best match each other. Using their method, the authors were able to achieve a between-subject classification accuracy on par with—and even greater than—within-subject accuracy.

Suppose that subject data are recorded in matrices $\mathbf{X}_{1:m} \in \mathbb{R}^{t \times n}$. This could be data from an experiment involving $m$ subjects, $t$ TRs, and $n$ voxels. We are interested in extending the regularized hyperalignment problem

$$
\begin{aligned}
\text{minimize} \quad & \sum_{i<j} \|\mathbf{X}_i \mathbf{R}_i - \mathbf{X}_j \mathbf{R}_j\|_{\mathrm{F}}^2 \\
\text{subject to} \quad & \mathbf{R}_k^T \mathbf{A}_k \mathbf{R}_k = \mathbf{I} \qquad k = 1, 2, \ldots, m \ ,
\end{aligned}
\tag{1}
$$

where matrices $\mathbf{A}_{1:m} \in \mathbb{R}^{n \times n}$ are symmetric and positive definite. In general, the above problem manifests itself in many application areas. For example, when $\mathbf{A}_k = \mathbf{I}$ we have hyperalignment or

a multi-set orthogonal Procrustes problem, commonly used in shape analysis [6, 7]. When $\mathbf{A}_k = \mathbf{X}_k^T \mathbf{X}_k$, (1) represents a form of multi-set Canonical Correlation Analysis (CCA) [12, 13, 8].

The success of hyperalignment engenders numerous questions and in this work we address two of them. First, is hyperalignment scalable? In [9], the authors consider a subset of ventral temporal cortex (VT), using hundreds of voxels. The relatively-low voxel count alleviates a huge computational cost and storage burden. However, the current method for solving (1) is infeasible when considering many or all voxels, and therefore limits the scope of hyperalignment to a local alignment procedure. For example, if $n = 50{,}000$ voxels, then storing the $n \times n$ matrix for one subject requires over 18 gigabytes of memory. Moreover, computing a *full* SVD for a matrix this size is a tall order.

Coupled with scalability, we also ask whether we can include new features of our subjects' data. For example, we may want to augment the input data with the associated second-order mixtures, i.e., $n$ voxels become $\binom{n}{1} + \binom{n}{2} = n(n+1)/2$ features. Again, for a reasonably-sized voxel count, running hyperalignment is infeasible.

Addressing scalability and feature extension results in the main contribution of *kernel hyperalignment*. The inclusion of a large feature space motivates the use of kernel methods. Additionally, numerous optimization problems that use the kernel trick possess global optimizers spanned by the mapped examples. This is guaranteed by the Representer Theorem [14, 18]. Therefore, the two separate issues of scalability and feature extension are merged into a single problem through the use of kernel methods. With kernel hyperalignment, the bottleneck shifts from voxel count to the number of TRs times subjects (or the original inputs to the number of examples).

The problem we address in this paper is the alignment of multiple datasets in the same and extended feature space. Multi-set data analysis by means of kernel methods has already been considered in the framework of CCA [16, 1]. Our approach deviates from [1] and [15] because we focus on alignment and never leave feature space until training and testing. We use the kernel trick as a means of navigating through a high-dimensional orthogonal group. Our CCA variant is more constrained, and each dataset is assigned the same kernel, supplying us with a richer, single reproducing kernel Hilbert space (RKHS) over a collection of $m$ smaller and distinct ones. Allowing for subject-specific kernels leads to the difficult problem of selecting them—a significantly harder problem than selecting a single kernel. In this respect, we assume a single kernel can provide the sought-after linearity used for comparing multiple datasets.

The paper is organized as follows: in §2 we review regularized hyperalignment, or the regularized multi-set orthogonal Procrustes problem. Next, in §3 we formulate its kernel variant, and in §4 we discuss classification with aligned data. We provided experimental results in §5, and we conclude in §6. All proofs are supplied in the Supplemental Material.

## 2  Hyperalignment

The hyperalignment problem of (1) is equivalent to [7]:

$$
\begin{aligned}
&\text{minimize} \quad \textstyle\sum_{i=1}^{m} \|\mathbf{X}_i \mathbf{R}_i - \mathbf{Y}\|_{\mathrm{F}}^2 \\
&\text{subject to} \quad \mathbf{Y} = \tfrac{1}{m} \textstyle\sum_{j=1}^{m} \mathbf{X}_j \mathbf{R}_j \quad \text{and} \quad \mathbf{R}_k^T \mathbf{A}_k \mathbf{R}_k = \mathbf{I} \ \text{for } k = 1, \ldots, m \ .
\end{aligned}
\tag{2}
$$

The matrix $\mathbf{Y}$ is the image centroid and serves as the catalyst for computing a solution: for dataset $i$, fix a centroid and solve for $\mathbf{R}_i$. This process cycles over all datasets for a specified number of rounds, or until approximate convergence is reached (see Algorithm 1). The dynamic centroid $\mathbf{Y}$ can be a sample mean or a leave-one-out (LOO) mean. Regardless of type, the last round should use the fixed sample mean provided by the penultimate round. We can set $\mathbf{Q}_k = \mathbf{A}_k^{1/2} \mathbf{R}_k$, using the symmetric, positive definite square root[1], yielding the key operation

$$
\begin{aligned}
&\text{minimize} \quad \|\mathbf{X}_k \mathbf{A}_k^{-\frac{1}{2}} \mathbf{Q}_k - \mathbf{Y}\|_{\mathrm{F}}^2 \\
&\text{subject to} \quad \mathbf{Q}_k^T \mathbf{Q}_k = \mathbf{I} \ .
\end{aligned}
\tag{3}
$$

The above is the familiar orthogonal Procrustes problem [19] and is solved using the SVD of $\mathbf{A}_k^{-\frac{1}{2}} \mathbf{X}_k^T \mathbf{Y}$.

# 3   Kernel Hyperalignment

The previous section dealt with alignment based on the original data. In the context of optimization, the alignment problem of (1) is indifferent to both data generation and data recording. There are, however, implicit assumptions about these two processes. The data are generated according to a common input signal, and each of the $m$ datasets represents a specific *view* of this signal. In other words, the matrices $\mathbf{X}_{1:m}$ have row correspondence. The alignment problem of (1) seeks column correspondence through a linear mapping of the original features.

In fMRI, the $m$ views are manifested by $m$ subjects experiencing a common, synchronous stimulus. Each data matrix records fMRI time-series data: the rows are indexed by a TR and the columns are indexed by a voxel. There are $t$ TRs and $n$ voxels per subject, i.e., $\mathbf{X}_k \in \mathbb{R}^{t \times n}$. The synchrony of the stimulus ensures row correspondence. Hyperalignment can be posed as the minimization problem of (2) with $\mathbf{A}_k = \mathbf{I}$. Voxel (column) correspondence is then achieved via an orthogonal constraint placed on each of the linear mappings. The orthogonal constraint present in hyperalignment follows a subject-independent isometry assumption. We can view the time-series data of each subject as a trajectory in $\mathbb{R}^n$. For a fixed stimulus this trajectory is [approximately] identical—up to a rotation-reflection—across subjects.

As stated above, we are assuming equivalence of the per-view information in its original form, but we are not assuming that this information can be related through a linear mapping. Now suppose there is a common set of $N$ features—derived from each $n$-dimensional example—that does allow for a linear relationship between views. Alternatively, there may be derivative features of interest that lead to better alignment via a linear mapping. For example, it is conceivable that second-order data, i.e., pairwise mixtures of the original data, obey a linear construct and may be a preferred feature set for alignment. In general, we wish to formulate an alignment technique for this new feature set. Rather than limit expression of the data to the $n$ given coordinates, we consider an $N$-coordinate representation, where $N$ may be much greater than $n$.

Let $\mathbf{X}_i \in \mathbb{R}^{t \times n}$ have $i'$-th row $[\mathbf{x}_{i'}^i]^T$ with $\mathbf{x}_{i'}^i \in \mathbb{R}^n$. We introduce the row-based mapping of $\mathbf{X}_i$:

$$\Phi(\mathbf{X}_i) = \begin{pmatrix} \phi_1(\mathbf{x}_1^i) & \phi_2(\mathbf{x}_1^i) & \cdots & \phi_N(\mathbf{x}_1^i) \\ \vdots & \vdots & & \vdots \\ \phi_1(\mathbf{x}_t^i) & \phi_2(\mathbf{x}_t^i) & \cdots & \phi_N(\mathbf{x}_t^i) \end{pmatrix} \in \mathbb{R}^{t \times N} . \tag{4}$$

The $N$ functions $\phi_{1:N} : \mathbb{R}^n \to \mathbb{R}$ are used to derive $N$ features from the original data. For matrix $\mathbf{X}_i \in \mathbb{R}^{t \times n}$ let $\mathbf{\Phi}_i = \Phi(\mathbf{X}_i)$. In general, for $\mathbf{X}_i \in \mathbb{R}^{t \times n}$ and $\mathbf{X}_j \in \mathbb{R}^{s \times n}$, we define the Gram matrix $\mathbf{K}_{ij} \triangleq \mathbf{\Phi}_i \mathbf{\Phi}_j^T \in \mathbb{R}^{t \times s}$. We also write $\mathbf{K}_i \triangleq \mathbf{K}_{ii} = \mathbf{\Phi}_i \mathbf{\Phi}_i^T$. We assume that there is an appropriate positive definite kernel, $\hat{k} : \mathbb{R}^n \times \mathbb{R}^n \to \mathbb{R}$, so that we can leverage the kernel trick [2, 10] and obtain the $i'j'$-th element of $\mathbf{K}_{ij}$ via

$$(\mathbf{K}_{ij})_{i'j'} = \hat{k}(\mathbf{x}_{i'}^i, \mathbf{x}_{j'}^j) . \tag{5}$$

Using the feature map $\Phi(\cdot)$, we form the regularized *Kernel Hyperalignment* problem:

$$\begin{aligned} \text{minimize} \quad & \sum_{i<j} \|\Phi(\mathbf{X}_i)\mathbf{R}_i - \Phi(\mathbf{X}_j)\mathbf{R}_j\|_F^2 \\ \text{subject to} \quad & \mathbf{R}_k^T \mathbf{A}_k \mathbf{R}_k = \mathbf{I} \text{ for } k = 1, \dots, m . \end{aligned} \tag{6}$$

The latent variables are $\mathbf{R}_{1:m} \in \mathbb{R}^{N \times N}$ and we are given symmetric, positive definite matrices $\mathbf{A}_{1:m} \in \mathbb{R}^{N \times N}$. Although different than the original hyperalignment problem, obtaining a solution to (6) is accomplished in the same way: fix a centroid and find the individual linear maps. To this end, the key operation involves solving

$$\underset{\mathbf{R}^T \mathbf{A}_i \mathbf{R} = \mathbf{I}}{\arg \min} \|\mathbf{\Phi}_i \mathbf{R} - \mathbf{\Psi}\|_F^2 \qquad \text{or} \qquad \underset{\mathbf{Q}^T \mathbf{Q} = \mathbf{I}}{\arg \min} \|\mathbf{\Phi}_i \mathbf{A}_i^{-\frac{1}{2}} \mathbf{Q} - \mathbf{\Psi}\|_F^2 , \tag{7}$$

where $\mathbf{\Phi}_i = \Phi(\mathbf{X}_i)$, $i \geq 1$, is the current, individual dataset under consideration and $\mathbf{\Psi} = \frac{1}{|\mathcal{A}|} \sum_{j \in \mathcal{A}} \mathbf{\Phi}_j \hat{\mathbf{R}}_j$ is a centroid based on the current estimates of $\mathbf{R}_{1:m}$, denoted $\hat{\mathbf{R}}_{1:m}$. The index set $\mathcal{A} \subseteq \{1, \dots, m\}$ determines how the estimated centroid is calculated (sample or LOO mean).

The difficulty of (7) lies in the size of $N$. Any of the well-known kernels correspond to an $N$ so large that direct computation is generally impractical. For example, if using second-order interactions as the feature set, the number of unknowns in kernel hyperalignment is $O(mn^4)$ in contrast to $O(mn^2)$ unknowns for hyperalignment. Nevertheless, the minimization problem of (7) places us in familiar territory of solving an orthogonal Procrustes problem.

Since we are now in feature space, the matrix $\mathbf{A}_i$ poses a problem unless we confine it to a specific form. For example, if $\mathbf{A}_i$ is random, finding $\mathbf{A}_i^{-1/2}$ would be infeasible for large $N$. Additionally, the constraint $\mathbf{R}_i^T \mathbf{A}_i \mathbf{R}_i = \mathbf{I}$ would lack any intuition. Therefore, we restrict $\mathbf{A}_i = \alpha \mathbf{I} + \beta \mathbf{\Phi}_i^T \mathbf{\Phi}_i$ with $\alpha > 0$ and $\beta \geq 0$. As with regularized hyperalignment [22], when $(\alpha, \beta) = (1, 0)$ we obtain hyperalignment and when $(\alpha, \beta) \approx (0, 1)$ we obtain a form of CCA.

Let $\mathbf{K}_i$ have eigen-decomposition $\mathbf{V}_i \mathbf{\Lambda}_i \mathbf{V}_i^T$, where $\mathbf{\Lambda}_i = \mathbf{diag}\{\lambda_{i1}, \ldots, \lambda_{it}\}$ or $\mathbf{diag}_j\{\lambda_{ij}\}$ for short. We introduce two symmetric, positive definite matrices: $\mathbf{B}_i = \mathbf{V}_i \, \mathbf{diag}_j\{\frac{1}{\sqrt{\alpha + \beta \lambda_{ij}}}\} \mathbf{V}_i^T$ and $\mathbf{C}_i = \mathbf{V}_i \, \mathbf{diag}_j\{\frac{1}{\lambda_{ij}}(\frac{1}{\sqrt{\alpha + \beta \lambda_{ij}}} - \frac{1}{\sqrt{\alpha}})\} \mathbf{V}_i^T$.

**Lemma 3.1.** *For $\mathbf{A}_i = \alpha \mathbf{I} + \beta \mathbf{\Phi}_i^T \mathbf{\Phi}_i$ we have $\mathbf{A}_i^{-\frac{1}{2}} = \frac{1}{\sqrt{\alpha}} \mathbf{I} + \mathbf{\Phi}_i^T \mathbf{C}_i \mathbf{\Phi}_i$ and $\mathbf{\Phi}_i \mathbf{A}_i^{-\frac{1}{2}} = \mathbf{B}_i \mathbf{\Phi}_i$.*

We can use Lemma 3.1 to transform (7) into

$$\underset{\mathbf{Q}^T \mathbf{Q} = \mathbf{I}}{\arg\min} \|\mathbf{B}_i \mathbf{\Phi}_i \mathbf{Q} - \mathbf{\Psi}\|_{\mathrm{F}}^2 \qquad \text{or} \qquad \underset{\mathbf{Q}^T \mathbf{Q} = \mathbf{I}}{\arg\max} \operatorname{tr}\left(\mathbf{Q}^T \mathbf{\Phi}_i^T \mathbf{B}_i \left[\frac{1}{|\mathcal{A}|} \sum_{j \in \mathcal{A}} \mathbf{B}_j \mathbf{\Phi}_j \hat{\mathbf{Q}}_j\right]\right), \quad (8)$$

where $\hat{\mathbf{Q}}_j$ is the current estimate of $\mathbf{Q}_j$. Solving for the matrix $\mathbf{Q}$ is still well beyond practical computation. The following lemma is the gateway for managing this problem.

**Lemma 3.2.** *If $\tilde{\mathbf{U}} \in St(N, d)$ and $\tilde{\mathbf{G}} \in \mathcal{O}(d)$, then $\tilde{\mathbf{Q}} = \mathbf{I}_N - \tilde{\mathbf{U}}(\mathbf{I}_d - \tilde{\mathbf{G}})\tilde{\mathbf{U}}^T \in \mathcal{O}(N).$*[2]

Familiar applications of the above lemma include the identity matrix ($\tilde{\mathbf{G}} = \mathbf{I}_d$) and Householder reflections ($\tilde{\mathbf{G}} = -\mathbf{I}_d$). If $\tilde{\mathbf{G}}$ is block diagonal with $2 \times 2$ blocks of Givens rotations, then the columns of $\tilde{\mathbf{U}}$, taken two at a time, are the two-dimensional planes of rotation [7]. We therefore refer to $\tilde{\mathbf{U}}$ as the *plane support matrix*.

Lemma 3.2 can be interpreted as a lifting mechanism for identity deviations. The difference $\mathbf{I}_d - \tilde{\mathbf{G}}$ represents a $\mathcal{O}(d)$ deviation from identity. Applying $\tilde{\mathbf{U}}(\mathbf{I}_d - \tilde{\mathbf{G}})\tilde{\mathbf{U}}^T = \mathbf{I}_N - \tilde{\mathbf{Q}}$, "lifts" this difference to a $\mathcal{O}(N)$ deviation from identity. Reversing directions, we can also utilize Lemma 3.2 for compressing $\mathcal{O}(N)$. From $\mathbf{I}_N - \tilde{\mathbf{Q}} = \tilde{\mathbf{U}}(\mathbf{I}_d - \tilde{\mathbf{G}})\tilde{\mathbf{U}}^T$, the rank of the deviation, $\mathbf{I}_N - \mathbf{Q}$, is upper bounded by $d$, producing a subset of $\mathcal{O}(N)$.

Motivated by Lemma 3.2 we impose

$$\mathbf{Q}_i = \mathbf{I}_N - \mathbf{U}(\mathbf{I} - \mathbf{G}_i)\mathbf{U}^T , \qquad (9)$$

where $\mathbf{U} \in St(N, r)$, $\mathbf{G}_i \in \mathcal{O}(r)$, and $1 \leq r \leq N$. Ideally, we want $r$ small to benefit from a reduced dimension. As is typically the case when using kernel methods, leveraging the Representer Theorem shifts the dimensionality of the problem from the feature cardinality to the number of examples, i.e., $r = mt$. We pool all of the data, forming the $mt \times N$ matrix

$$\mathbf{\Phi}_0 = \begin{bmatrix} \mathbf{\Phi}_1^T & \mathbf{\Phi}_2^T & \cdots & \mathbf{\Phi}_m^T \end{bmatrix}^T , \qquad (10)$$

and set $\mathbf{U} = \mathbf{\Phi}_0^T \mathbf{K}_0^{-\frac{1}{2}} \in \mathbb{R}^{N \times r}$ with $\mathbf{K}_0 = \mathbf{\Phi}_0 \mathbf{\Phi}_0^T$ assumed positive definite. As long as $r \leq N$, the orthogonality constraint is met because $(\mathbf{\Phi}_0^T \mathbf{K}_0^{-\frac{1}{2}})^T (\mathbf{\Phi}_0^T \mathbf{K}_0^{-\frac{1}{2}}) = \mathbf{K}_0^{-\frac{1}{2}} \mathbf{K}_0 \mathbf{K}_0^{-\frac{1}{2}} = \mathbf{I}_r$.

**Theorem 3.3** (Hyperalignment Representer Theorem). *Within the set of global minimizers of* (6) *there exists a solution $\{\mathbf{R}_1^\star, \ldots, \mathbf{R}_m^\star\} = \{\mathbf{A}_1^{-\frac{1}{2}} \mathbf{Q}_1^\star, \ldots, \mathbf{A}_m^{-\frac{1}{2}} \mathbf{Q}_m^\star\}$ that admits a representation $\mathbf{Q}_i^\star = \mathbf{I}_N - \mathbf{U}(\mathbf{I} - \mathbf{G}_i^\star)\mathbf{U}^T$, where $\mathbf{U} = \mathbf{\Phi}_0^T \mathbf{K}_0^{-\frac{1}{2}}$ and $\mathbf{G}_i^\star \in \mathcal{O}(mt)$ $(i = 1, \ldots, m)$.*

**Input**: $\mathbf{X}_{1:m} \in \mathbb{R}^{t \times n}$, $\mathbf{A}_{1:m} \in \mathbb{R}^{n \times n}$
**Output**: $\mathbf{R}_{1:m} \in \mathbb{R}^{n \times n}$
Initialize $\mathbf{Q}_{1:m}$ as identity $(n \times n)$
Set $\tilde{\mathbf{X}}_i \overset{1:m}{\longleftarrow} \mathbf{X}_i \mathbf{A}_i^{-1/2}$
**foreach** *round* **do**
    **foreach** *subject/view i* **do**
        $\mathcal{A} \leftarrow \begin{cases} \{1, 2, \ldots, m\} & \text{sample mean} \\ \{1, 2, \ldots, m\} \setminus \{i\} & \text{LOO mean} \end{cases}$

        $\mathbf{Y} \leftarrow \dfrac{1}{|\mathcal{A}|} \sum_{j \in \mathcal{A}} \tilde{\mathbf{X}}_j \mathbf{Q}_j$
        $[\bar{\mathbf{U}} \ \bar{\boldsymbol{\Sigma}} \ \bar{\mathbf{V}}] \leftarrow \mathrm{SVD}(\tilde{\mathbf{X}}_i^T \mathbf{Y})$
        $\mathbf{Q}_i \leftarrow \bar{\mathbf{U}} \bar{\mathbf{V}}^T$
    **end**
**end**
**foreach** *subject/view i* **do**
    $\mathbf{R}_i \leftarrow \mathbf{A}_i^{-\frac{1}{2}} \mathbf{Q}_i$
**end**

**Algorithm 1:** Regularized Hyperalignment

**Input**: $\hat{k}(\cdot, \cdot)$, $\alpha$, $\beta$, $\mathbf{X}_{1:m} \in \mathbb{R}^{t \times n}$
**Output**: $\mathbf{R}_{1:m}$, linear maps in feature space
Initialize feature maps $\boldsymbol{\Phi}_1, \ldots, \boldsymbol{\Phi}_m \in \mathbb{R}^{t \times N}$
Initialize plane support $\boldsymbol{\Phi}_0 = \begin{bmatrix} \boldsymbol{\Phi}_1^T & \boldsymbol{\Phi}_2^T & \cdots & \boldsymbol{\Phi}_m^T \end{bmatrix}^T$
Initialize $\mathbf{G}_{1:m} \in \mathbb{R}^{r \times r}$ as identity $(r = mt)$
**foreach** *round* **do**
    **foreach** *subject/view i* **do**
        $\mathcal{A} \leftarrow \begin{cases} \{1, 2, \ldots, m\} & \text{sample mean} \\ \{1, 2, \ldots, m\} \setminus \{i\} & \text{LOO mean} \end{cases}$

        $\mathbf{Y} \leftarrow \dfrac{1}{|\mathcal{A}|} \sum_{j \in \mathcal{A}} \tilde{\mathbf{B}}_j \mathbf{G}_j$
        $[\bar{\mathbf{U}} \ \bar{\boldsymbol{\Sigma}} \ \bar{\mathbf{V}}] \leftarrow \mathrm{SVD}(\tilde{\mathbf{B}}_i^T \mathbf{Y})$
        $\mathbf{G}_i \leftarrow \bar{\mathbf{U}} \bar{\mathbf{V}}^T$
    **end**
**end**
**foreach** *subject/view i* **do**
    $\mathbf{Q}_i \leftarrow \mathbf{I} - \boldsymbol{\Phi}_0^T \mathbf{K}_0^{-\frac{1}{2}} (\mathbf{I}_r - \mathbf{G}_i) \mathbf{K}_0^{-\frac{1}{2}} \boldsymbol{\Phi}_0$
    $\mathbf{R}_i \leftarrow \mathbf{A}_i^{-\frac{1}{2}} \mathbf{Q}_i$
**end**

**Algorithm 2:** Regularized Kernel Hyperalignment

When $mt$ is large enough so that evaluating an SVD of numerous $mt \times mt$ matrices is prohibitive, we can first perform PCA-like reduction. Let $\mathbf{K}_0$ have eigen-decomposition $\mathbf{V}_0 \boldsymbol{\Lambda}_0 \mathbf{V}_0^T$, where the nonnegative diagonal entries of $\boldsymbol{\Lambda}_0$ are sorted in decreasing order. We set $\boldsymbol{\Phi}_{0'} = \mathbf{V}_{0'}^T \boldsymbol{\Phi}_0$, where $\mathbf{V}_{0'}$ is formed by the first $r$ columns of $\mathbf{V}_0$, and then use $\mathbf{U} = \boldsymbol{\Phi}_{0'}^T \mathbf{K}_{0'}^{-1/2}$. In general, rather than compute $\mathbf{Q}$ according to (7), involving $N(N-1)/2 = O(N^2)$ degrees of freedom (when $N$ is finite), we end up with $r(r-1)/2 = O(r^2)$ degrees of freedom via the kernel trick.

Let $\tilde{\mathbf{B}}_i = \mathbf{B}_i \mathbf{K}_{i0} \mathbf{K}_0^{-\frac{1}{2}} \in \mathbb{R}^{t \times r}$. We reduce (8) in terms of $\mathbf{G}_i$ and obtain (Supplementary Material)

$$\mathbf{G}_i = \underset{\mathbf{G} \in \mathcal{O}(r)}{\arg\max} \ \mathrm{tr} \left( \mathbf{G}^T \tilde{\mathbf{B}}_i^T \left[ \frac{1}{|\mathcal{A}|} \sum_{j \in \mathcal{A}} \tilde{\mathbf{B}}_j \hat{\mathbf{G}}_j \right] \right), \tag{11}$$

where $\hat{\mathbf{G}}_j$ is the current estimate of $\mathbf{G}_j$. Equation (11) is the classical orthogonal Procrustes problem. If $\bar{\mathbf{U}} \bar{\boldsymbol{\Sigma}} \bar{\mathbf{V}}^T$ is the SVD of $\mathbf{G}^T \tilde{\mathbf{B}}_i^T \left[ \frac{1}{|\mathcal{A}|} \sum_{j \in \mathcal{A}} \tilde{\mathbf{B}}_j \hat{\mathbf{G}}_j \right]$, then a maximizer is given by $\bar{\mathbf{U}} \bar{\mathbf{V}}^T$ [7]. The kernel hyperalignment procedure is given in Algorithm 2. Using the approach taken in this section also leads to an efficient solution of the standard orthogonal Procrustes problem for $n \geq 2t$ (Supplementary Material). In turn, this leads to an efficient iterative solution for the hyperalignment problem when $n$ is large.

## 4 Alignment Assessment

An alignment procedure is not subject to the typical train-and-test paradigm. The lack of spatial correspondence demands an align-train-test approach. We assume these three sets have within-subject (or within-view) alignment. With all other parameters fixed, if the aligned test error is smaller than the unaligned test error, there is strong evidence suggesting that alignment was the underlying cause.

Kernel hyperalignment returns linear transformations $\mathbf{R}_{1:m}$ that act on data living in feature space. In general, we cannot directly train and test in the feature space due to its large size. We can, however, learn from relational data. For example, we can compute distances between examples and, subsequently, produce nearest neighbor classifiers. Assume $(\alpha, \beta) = (1, 0)$, i.e., the $\mathbf{R}_{1:m}$

are orthogonal. If $\mathbf{x}_1 \in \mathbb{R}^n$ is a view-$i$ example and $\mathbf{x}_2 \in \mathbb{R}^n$ is a view-$j$ example, the respective pre-aligned and post-aligned squared distances between the two examples are given by

$$\|\Phi(\mathbf{x}_1^T) - \Phi(\mathbf{x}_2^T)\|_{\mathrm{F}}^2 = \hat{k}(\mathbf{x}_1, \mathbf{x}_1) + \hat{k}(\mathbf{x}_2, \mathbf{x}_2) - 2\hat{k}(\mathbf{x}_1, \mathbf{x}_2) \qquad (12)$$

$$\|\Phi(\mathbf{x}_1^T)\mathbf{R}_i - \Phi(\mathbf{x}_2^T)\mathbf{R}_j\|_{\mathrm{F}}^2 = \hat{k}(\mathbf{x}_1, \mathbf{x}_1) + \hat{k}(\mathbf{x}_2, \mathbf{x}_2) - 2\Phi(\mathbf{x}_1^T)\mathbf{R}_i\mathbf{R}_j^T\Phi(\mathbf{x}_2^T)^T \ . \qquad (13)$$

The cross-term in (13) has not been expanded for a simple reason: it is too messy. We realized early on that the alignment and training phase would be replete with lengthy expansions and, consequently, sought to simplify matters with a computer science solution. Both binary and unary operations in feature space can be accomplished with a simple class. Our `Phi` class stores expressions of the following forms:

$$\underbrace{\sum_{k=1}^{K}\mathbf{M}_k\Phi(\mathbf{X}_{\overline{a}(k)})}_{\text{Type 1}} \qquad \underbrace{\sum_{k=1}^{K}\Phi(\mathbf{X}_{\underline{a}(k)})^T\mathbf{M}_k}_{\text{Type 2}} \qquad \underbrace{b\mathbf{I}_N + \sum_{k=1}^{K}\Phi(\mathbf{X}_{\underline{a}(k)})^T\mathbf{M}_k\Phi(\mathbf{X}_{\overline{a}(k)})}_{\text{Type 3}} . \qquad (14)$$

Each class instance stores matrices $\mathbf{M}_{1:K}$, scalar $b$, right address vector $\overline{a}$, and left address vector $\underline{a}$. The address vectors are pointers to the input data. This allows for faster manipulation and smaller memory allocation. Addition and subtraction require a common type. If types match, then the $\mathbf{M}$ matrices must be checked for compatible sizes. Multiplication is performed for types 1 with 2, 1 with 3, 2 with 1, 3 with 2, and 3 with 3. The first of these cases, for example, produces a numeric result via the kernel trick. We also define scalar multiplication and division for all types and matrix multiplication for types 1 and 2. A transpose operator applies for all types and maps type 1 to 2, 2 to 1, and 3 to 3. More advanced operations, such as powers and inverses, are also possible. Our implementation was done in Matlab.

The construction of the `Phi` class allows us to stay in feature space and avoid lengthy expansions. In turn, this facilitates implementing the richer set of SVM classifiers. Let $\mathbf{X}_{\overline{1}}, \ldots, \mathbf{X}_{\overline{m}} \in \mathbb{R}^{s \times n}$ be our training data with feature representation $\boldsymbol{\Phi}_{\overline{\imath}} = \Phi(\mathbf{X}_{\overline{\imath}}) \in \mathbb{R}^{s \times N}$. Recall that kernel hyperalignment seeks to align in feature space. Before alignment we might have considered $\mathbf{K}_{\overline{\imath}\overline{\jmath}} = \boldsymbol{\Phi}_{\overline{\imath}}\boldsymbol{\Phi}_{\overline{\jmath}}^T$; we now consider the Gram matrix $(\boldsymbol{\Phi}_{\overline{\imath}}\mathbf{R}_i)(\boldsymbol{\Phi}_{\overline{\jmath}}\mathbf{R}_j)^T = \boldsymbol{\Phi}_{\overline{\imath}}\mathbf{R}_i\mathbf{R}_j^T\boldsymbol{\Phi}_{\overline{\jmath}}^T$. If every row of $\mathbf{X}_{\overline{\imath}}$ has a corresponding label, we can train an SVM with

$$\mathbf{K}_{\bar{A}} = \begin{pmatrix} \boldsymbol{\Phi}_{\overline{1}}\mathbf{R}_1 \\ \vdots \\ \boldsymbol{\Phi}_{\overline{m}}\mathbf{R}_m \end{pmatrix} \times \begin{pmatrix} \boldsymbol{\Phi}_{\overline{1}}\mathbf{R}_1 \\ \vdots \\ \boldsymbol{\Phi}_{\overline{m}}\mathbf{R}_m \end{pmatrix}^T = \begin{pmatrix} \boldsymbol{\Phi}_{\overline{1}}\mathbf{R}_1\mathbf{R}_1^T\boldsymbol{\Phi}_{\overline{1}}^T & \boldsymbol{\Phi}_{\overline{1}}\mathbf{R}_1\mathbf{R}_2^T\boldsymbol{\Phi}_{\overline{2}}^T & \cdots & \boldsymbol{\Phi}_{\overline{1}}\mathbf{R}_1\mathbf{R}_m^T\boldsymbol{\Phi}_{\overline{m}}^T \\ \boldsymbol{\Phi}_{\overline{2}}\mathbf{R}_2\mathbf{R}_1^T\boldsymbol{\Phi}_{\overline{1}}^T & \boldsymbol{\Phi}_{\overline{2}}\mathbf{R}_2\mathbf{R}_2^T\boldsymbol{\Phi}_{\overline{2}}^T & & \\ \vdots & & \ddots & \\ \boldsymbol{\Phi}_{\overline{m}}\mathbf{R}_m\mathbf{R}_1^T\boldsymbol{\Phi}_{\overline{1}}^T & & & \boldsymbol{\Phi}_{\overline{m}}\mathbf{R}_m\mathbf{R}_m^T\boldsymbol{\Phi}_{\overline{m}}^T \end{pmatrix} , \qquad (15)$$

where $\mathbf{K}_{\bar{A}} = \mathbf{K}_{\bar{A}}^T \in \mathbb{R}^{ms \times ms}$ denotes the aligned kernel matrix. The unaligned kernel matrix, $\mathbf{K}_{\bar{U}}$, is also an $m \times m$ block matrix with $ij$-th block $\mathbf{K}_{\overline{\imath}\overline{\jmath}}$.

Using the dual formulation of an SVM, a classifier can be constructed from the relational data exhibited among the examples [4]. Similar to a $k$-nearest neighbor classifier relying on pairwise distances, an SVM relies on the kernel matrix. The kernel matrix is a matrix of inner products and is therefore linear. This enables us to assess a partition-based alignment.

In fMRI, we perform two alignments—one for each hemisphere. Each alignment produces two aligned kernel matrices, which we sum and then input into an SVM. Thus, linearity provides us the means to handle finer partitions by simply summing the aligned kernel matrices.

Table 1: *Seven label classification using movie-based alignment*  Below is the cross-validated, between-subject classification accuracy (within-subject in brackets) with $(\alpha, \beta) = (1, 0)$.  Four hundred TRs per subject were used for the alignment. Chance $= 1/7 \approx 14.29\%$.

| Kernel | Ventral Temporal 2,997 voxels/hemisphere | | Entire Cortex 133,590 voxels/hemisphere | |
|---|---|---|---|---|
| | Anatomical | Kernel Hyp. | Anatomical | Kernel Hyp. |
| Linear | 35.71% [42.68%] | 48.57% [42.68%] | 34.64% [26.79%] | 36.25% [26.79%] |
| Quadratic | 35.00% [43.32%] | 50.36% [42.32%] | 36.07% [25.54%] | 36.43% [25.54%] |
| Gaussian | 36.25% [43.39%] | 48.57% [43.39%] | 36.07% [26.07%] | 36.43% [26.07%] |
| Sigmoid | 35.89% [43.21%] | 48.21% [43.21%] | 35.00% [26.79%] | 36.25% [26.79%] |

# 5  Experiments

The data used in this section consisted of fMRI time-series data from 10 subjects who viewed a movie and also engaged in a block-design visualization experiment [17]. Each subject saw *Raiders of the Lost Ark* (1981) lasting a total of 2213 TRs.  In the visualization experiment, subjects were shown images belonging to a specific class for 16 TRs followed by 10 TRs of rest.  The 7 classes were: (1) female face, (2) male face, (3) monkey, (4) house, (5) chair, (6) shoe and (7) dog.  There were 8 runs total, and each run had every image class represented once.

We assess alignment by classification accuracy. To provide the same number of voxels per ROI for all subjects, we first performed anatomical alignment. We then selected a contiguous block of 400 TRs from the movie data to serve as the per-subject input of the kernel hyperalignment. Next, we extracted labeled examples from the visualization experiment by taking an offset time average of each 16 TR class exposure.  An offset of 6 seconds factored in the hemodynamic response. This produced 560 labeled examples: 10 subjects $\times$ 8 runs/subject $\times$ 7 examples/run.

Kernel hyperalignment allows us to (a) use nonlinear measures of similarity, and (b) consider more voxels for the alignment. Consequently, we (a) experiment with a variety of kernels, and (b) do not need to pre-select or screen voxels as was done in [9]—we include them all. Table 1 features results from a 7-label classification experiment. Recall that a linear kernel reduces to hyperalignment. We classified using a multi-label $\nu$-SVM [3]. We used the first 400 TRs from each subject's movie data, and aligned each hemisphere separately.  The kernel functions are supplied in the Supplementary Material. As observed in [9] and repeated here, hyperalignment leads to increased between-subject accuracy and outperforms within-subject accuracy. Thus, we are extracting more common structure across subjects. Whereas employing Algorithm 1 for 2,997 voxels is feasible (and slow), 133,590 voxels is not feasible at all.

To complete the picture, we plot the effects of regularization. Figure 1 displays the cross-validated, between-subject classification accuracy for varying $(\alpha, \beta)$ where $\alpha = 1 - \beta$. This traces out a route from CCA ($\alpha \approx 0$) to hyperalignment ($\alpha = 1$). When compared to the alignments in [9], our voxel counts are orders of magnitude larger. For our four chosen kernels, hyperalignment ($\alpha = 1$) presents itself as the option with near-greatest accuracy.

Our results support the robustness of hyperalignment and imply that voxel selection may be a crucial pre-processing step when dealing with the whole volume.  More voxels mean more noisy voxels, and hyperalignment does not distinguish itself from anatomical alignment when the entire cortex is considered. We can visualize this phenomenon with Multidimensional Scaling (MDS) [21].

MDS takes as input all of the pairwise distances between subjects (the previous section discussed distance calculations). Figure 2 depicts the optimal Euclidean representation of our 10 subjects before and after kernel hyperalignment ($(\alpha, \beta) = (1, 0)$) with respect to the first 400 TRs of the movie data. Focusing on VT, kernel hyperalignment manages to cluster 7 of the 10 subjects. However, when we shift to the entire cortex, we see that anatomical alignment has already succeeded in a similar clustering. Kernel hyperalignment manages to group the subjects closer together, and manifests itself as a re-centering.

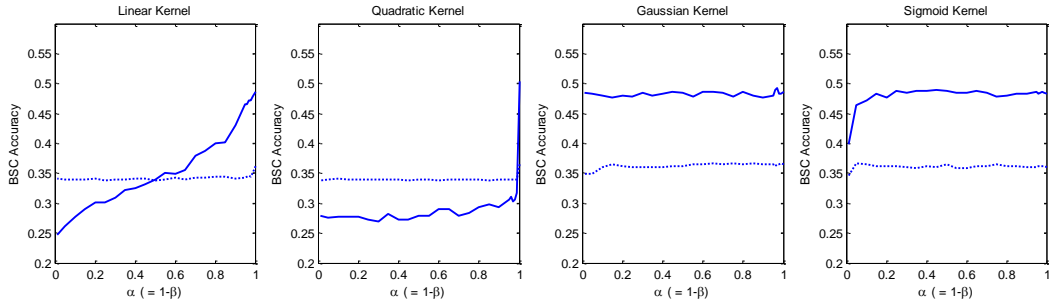

Figure 1: Cross-validated between-subject classification accuracy (7 labels) as a function of the regularization parameter, $\alpha = 1-\beta$, for various kernels after alignment. The solid curves are for Ventral Temporal and the dashed curves are for the entire cortex. Chance $= 1/7 \approx 14.29\%$.

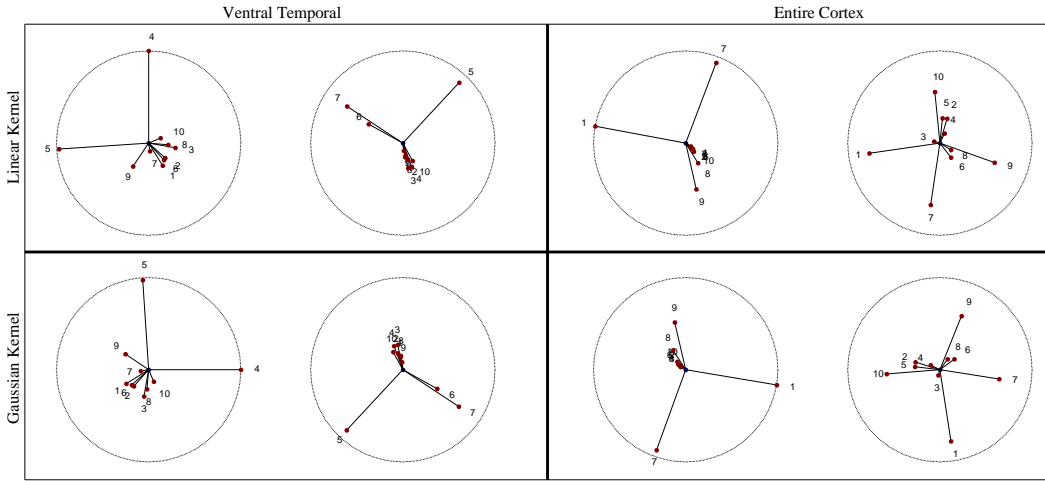

Figure 2: *Visualizing alignment with MDS* Each locus pair approximates the normalized relationship among the 10 subjects in 2D - before (left) and after (right) applying kernel hyperalignment. Centroids are translated to the origin and numbers correspond to individual subjects.

## 6 Conclusion

We have extended hyperalignment in both scale and feature space. Kernel hyperalignment can handle a large number of original features and incorporate nonlinear measures of similarity. We have also shown how to use the linear maps—applied in feature space—for post-alignment classification.

In the setting of fMRI, we have demonstrated successful alignment with a variety of kernels. Kernel hyperalignment achieved better between-subject classification over anatomical alignment for VT. There was no noticeable difference when we considered the entire cortex. Nevertheless, kernel hyperalignment proved robust and did not degrade with increasing voxel count.

We envision a fruitful path for kernel hyperalignment. Empirically, we have noticed a tradeoff between feature cardinality and classification accuracy, motivating the need for intelligent feature selection within our established framework. Although we have limited our focus to fMRI data analysis, kernel hyperalignment can be applied to other research areas which rely on multi-set Procrustes problems.

## Footnotes

[1] In practice, we would use the Cholesky factorization of $\mathbf{A}_k$. However, in deriving the kernel hyperalignment procedure it is necessary to familiarize the reader with this approach.

[2] $St(N, d) \triangleq \{\mathbf{Z} : \mathbf{Z} \in \mathbb{R}^{N \times d}, \mathbf{Z}^T \mathbf{Z} = \mathbf{I}_d\}$ is the $(N, d)$ Stiefel Manifold ($N \geq d$), and $\mathcal{O}(N) \triangleq \{\mathbf{Z} : \mathbf{Z} \in \mathbb{R}^{N \times N}, \mathbf{Z}^T \mathbf{Z} = \mathbf{I}_N\}$ is the orthogonal group of $N \times N$ matrices.

# References

[1] F.R. Bach and M.I. Jordan. Kernel independent component analysis. *The Journal of Machine Learning Research*, 3:1–48, 2003.

[2] C.M. Bishop. *Pattern Recognition and Machine Learning*. Springer, 2006.

[3] C.C. Chang and C.J. Lin. LIBSVM: A library for support vector machines. *ACM Transactions on Intelligent Systems and Technology*, 2:27:1–27:27, 2011. Software available at `http://www.csie.ntu.edu.tw/~cjlin/libsvm`.

[4] P.H. Chen, C.J. Lin, and B. Schölkopf. A tutorial on $\nu$-support vector machines. *Applied Stochastic Models in Business and Industry*, 21(2):111–136, 2005.

[5] A. Edelman, T. As, A. Arias, and T. Smith. The geometry of algorithms with orthogonality constraints. *SIAM J. Matrix Anal. Appl*, 1998.

[6] C. Goodall. Procrustes methods in the statistical analysis of shape. *Journal of the Royal Statistical Society. Series B (Methodological)*, pages 285–339, 1991.

[7] J.C. Gower and G.B. Dijksterhuis. *Procrustes Problems*, volume 30. Oxford University Press, USA, 2004.

[8] D.R. Hardoon, S. Szedmak, and J. Shawe-Taylor. Canonical correlation analysis: An overview with application to learning methods. *Neural Computation*, 16(12):2639–2664, 2004.

[9] J.V. Haxby, J.S. Guntupalli, A.C. Connolly, Y.O. Halchenko, B.R. Conroy, M.I. Gobbini, M. Hanke, and P.J. Ramadge. A common, high-dimensional model of the representational space in human ventral temporal cortex. *Neuron*, 72(2):404–416, 2011.

[10] T. Hofmann, B. Schölkopf, and A.J. Smola. Kernel methods in machine learning. *The Annals of Statistics*, pages 1171–1220, 2008.

[11] R.A. Horn and C.R. Johnson. *Matrix Analysis*. Cambridge University Press, 1990.

[12] H. Hotelling. Relations between two sets of variates. *Biometrika*, 28(3/4):321–377, 1936.

[13] J.R. Kettenring. Canonical analysis of several sets of variables. *Biometrika*, 58(3):433, 1971.

[14] G.S. Kimeldorf and G. Wahba. A correspondence between Bayesian estimation on stochastic processes and smoothing by splines. *The Annals of Mathematical Statistics*, 41(2):495–502, 1970.

[15] M. Kuss and T. Graepel. The geometry of kernel canonical correlation analysis. Technical report, Max Planck Institute, 2003.

[16] P.L. Lai and C. Fyfe. Kernel and nonlinear canonical correlation analysis. *International Journal of Neural Systems*, 10(5):365–378, 2000.

[17] M.R. Sabuncu, B.D. Singer, B. Conroy, R.E. Bryan, P.J. Ramadge, and J.V. Haxby. Function based inter-subject alignment of human cortical anatomy. *Cerebral Cortex*, 2009.

[18] B. Schölkopf, R. Herbrich, and A. Smola. A generalized representer theorem. In *Computational learning theory*, pages 416–426. Springer, 2001.

[19] P.H. Schonemann. A generalized solution of the orthogonal procrustes problem. *Psychometrika*, 31(1):1–10, March 1966.

[20] J. Talairach and P. Tournoux. *Co-planar stereotaxic atlas of the human brain: 3-dimensional proportional system: an approach to cerebral imaging*. Thieme, 1988.

[21] J.B. Tenenbaum, V. De Silva, and J.C. Langford. A global geometric framework for nonlinear dimensionality reduction. *Science*, 290(5500):2319–2323, 2000.

[22] H. Xu, A. Lorbert, P. J. Ramadge, J. S. Guntupalli, and J. V. Haxby. Regularized hyperalignment of multi-set fmri data. *Proceedings of the 2012 IEEE Signal Processing Workshop, Ann Arbor Michigan*, 2012.

